# Evidence-Specific Structures for Rich Tractable CRFs

**Anton Chechetka**
Carnegie Mellon University
antonc@cs.cmu.edu

**Carlos Guestrin**
Carnegie Mellon University
guestrin@cs.cmu.edu

## Abstract

We present a simple and effective approach to learning tractable conditional random fields with structure that depends on the evidence. Our approach retains the advantages of tractable discriminative models, namely efficient exact inference and arbitrarily accurate parameter learning in polynomial time. At the same time, our algorithm does not suffer a large expressive power penalty inherent to fixed tractable structures. On real-life relational datasets, our approach matches or exceeds state of the art accuracy of the dense models, and at the same time provides an order of magnitude speedup.

## 1 Introduction

Conditional random fields (CRFs, [1]) have been successful in modeling complex systems, with applications from speech tagging [1] to heart motion abnormality detection [2]. A key advantage of CRFs over other probabilistic graphical models (PGMs, [3]) stems from the observation that in almost all applications, some variables are unknown at test time (we will denote such variables $\mathcal{X}$), but others, called the *evidence* $\mathcal{E}$, *are known at test time.* While other PGM formulations model the joint distribution $P(\mathcal{X}, \mathcal{E})$, CRFs directly model conditional distributions $P(\mathcal{X} \mid \mathcal{E})$.

The discriminative approach adopted by CRFs allows for better approximation quality of the learned conditional distribution $P(\mathcal{X} \mid \mathcal{E})$, because the representational power of the model is not "wasted" on modeling $P(\mathcal{E})$. However, the better approximation comes at a cost of increased computational complexity for both structure [4] and parameter learning [1] as compared to generative models. In particular, unlike Bayesian networks or junction trees [3], (a) the likelihood of a CRF structure *does not decompose* into a combination of small subcomponent scores, making many existing approaches to structure learning inapplicable, and, (b) instead of computing optimal parameters in *closed form,* with CRFs one has to resort to gradient-based methods. Moreover, computing the gradient of the log-likelihood with respect to the CRF parameters requires inference in the current model to be done for every training datapoint. For high-treewidth models, even approximate inference is NP-hard [5].

To overcome the extra computational challenges posed by the conditional random fields, practitioners usually resort to several of the following approximations throughout the process:

- CRF structure is specified by hand, leading to suboptimal structures.
- Approximate inference during parameter learning results in suboptimal parameters.
- Approximate inference at test time results in suboptimal results [5].
- Replacing the CRF conditional likelihood objective with a more tractable one (e.g. [6]) results in suboptimal models (both in terms of learned structure and parameters).

Not only do all of the above approximation techniques lack any quality guarantees, but also combining several of them in the same system serves to further compound the errors.

A well-known way to avoid approximations in CRF parameter learning is to restrict the models to have *low treewidth*, where the dependencies between the variables $\mathcal{X}$ have a tree-like structure. For

such models, parameter learning and inference can be done exactly[1]; only structure learning involves approximations. The important dependencies between the variables $\mathcal{X}$, however, usually cannot *all* be captured with a *single* tree-like structure, so low-treewidth CRFs are rarely used in practice.

In this paper, we argue that it is the commitment to a *single* CRF structure *irrespective of the evidence* $\mathcal{E}$ that makes tree-like CRFs an inferior option. We show that tree CRFs with evidence-dependent structure, learned by a generalization of the Chow-Liu algorithm [7], (a) yield results equal to or significantly better than densely-connected CRFs on real-life datasets, and (b) are an order of magnitude faster than the dense models. More specifically, our contributions are as follows:

- Formally define CRFs with evidence-specific (ES) structure.
- Observe that, given the ES structures, CRF feature weights can be learned exactly.
- Generalize the Chow-Liu algorithm [7] to learn evidence-specific structures for tree CRFs.
- Generalize tree CRFs with evidence-specific structure (ESS-CRFs) to the relational setting.
- Demonstrate empirically the superior performance of ESS-CRFs over densely connected models in terms of both accuracy and runtime on real-life relational models.

## 2  Conditional random fields

A conditional random field with pairwise features[2] defines a conditional distribution $P(\mathcal{X}\,|\,\mathcal{E})$ as

$$P(\mathcal{X}\mid\mathcal{E}) = Z^{-1}(\mathcal{E})\exp\left\{\sum\nolimits_{(i,j)\in T}\sum\nolimits_{k}w_{ijk}f_{ijk}(X_i,X_j,\mathcal{E})\right\}, \qquad (1)$$

where functions $f$ are called *features,* $w$ are feature weights, $Z(\mathcal{E})$ is the normalization constant (which depends on evidence), and $T$ is the set of edges of the model. To reflect the fact that $P(\mathcal{X}\mid\mathcal{E})$ depends on the weights $w$, we will write $P(\mathcal{X}|\mathcal{E},w)$. To apply a CRF model, one first defines the set of features $f$. A typical feature may mean that two pixels $i$ and $j$ in the same image segment tend to have have similar colors: $f(X_i,X_j,\mathcal{E})\equiv\mathbb{I}(X_i=X_j,|\text{color}_i-\text{color}_j|<\delta)$, where $\mathbb{I}(\cdot)$ is an indicator function. Given the features $f$ and training data $\mathcal{D}$ that consists of fully observed assignments to $\mathcal{X}$ and $\mathcal{E}$, the optimal feature weights $w^*$ maximize the conditional log-likelihood (CLLH) of the data:

$$w^*=\arg\max\sum_{(\mathbf{X},\mathbf{E})\in\mathcal{D}}\log P(\mathbf{X}\,|\,\mathbf{E},w) = \arg\max\sum_{(\mathbf{X},\mathbf{E})\in\mathcal{D}}\left(\sum_{(i,j)\in T,k}w_{ijk}f_{ijk}(X_i,X_j,\mathbf{E}) - \log Z(\mathbf{E},w)\right). \quad (2)$$

The problem (2) does not have a closed form solution, but has a unique global optimum that can be found using any gradient-based optimization technique because of the following fact [1]:

**Fact 1** *Conditional log-likelihood* (2)*, abbreviated CLLH, is concave in* $w$. *Moreover,*

$$\frac{\partial\log P(\mathbf{X}\,|\,\mathbf{E},w)}{\partial w_{ijk}} = f_{ijk}(\mathbf{X_i},\mathbf{X_j},\mathbf{E}) - \mathbb{E}_{P(X_i,X_j|\mathbf{E},w)}\left[f_{ijk}(X_i,X_j,\mathbf{E})\right], \qquad (3)$$

*where* $\mathbb{E}_P$ *denotes expectation with respect to a distribution* $P$.

Convexity of the negative CLLH objective and the closed-form expression for the gradient lets us use convex optimization techniques such as L-BFGS [9] to find the unique optimum $w^*$. However, the gradient (3) contains the conditional distribution over $X_iX_j$, so computing (3) requires inference in the model for every datapoint. Time complexity of the exact inference is exponential in the treewidth of the graph defined by edges $T$ [5]. Therefore, exact evaluation of the CLLH objective (2) and gradient (3) and exact inference at test time are all only feasible for models with low-treewidth $T$.

Unfortunately, restricting the space of models to only those with low treewidth severely decreases the expressive power of CRFs. Complex dependencies of real-life distributions usually cannot be adequately captured by a single tree-like structure, so most of the models used in practice have high treewidth, making exact inference infeasible. Instead, approximate inference techniques, such as

belief propagation [10, 11] or sampling [12] are used for parameter learning and at test time. Approximate inference is NP-hard [5], so approximate inference algorithms have very few result quality guarantees. Greater expressive power of the models is thus obtained at the expense of worse quality of estimated parameters and inference. Here, we show an alternative way to increase expressive power of tree-like structured CRFs *without* sacrificing optimal weights learning and exact inference at test time. In practice, our approach is much better suited for relational than for propositional settings, because of much higher parameters dimensionality in the propositional case. However, we first present in detail the propositional case theory to better convey the key high-level ideas.

## 3 Evidence-specific structure for CRFs

Observe that, given a particular evidence value $\mathbf{E}$, the set of edges $T$ in the CRF formulation (1) actually can be viewed as a *supergraph* of the conditional model over $\mathcal{X}$. An edge $(r, s) \in T$ can be "disabled" in the following sense: if for $\mathcal{E} = \mathbf{E}$ the edge features are identically zero, $f_{rsk}(X_r, X_s, \mathbf{E}) \equiv 0$, regardless of the values of $X_r$ and $X_s$, then

$$\sum_{(i,j) \in T} \sum_k w_{ijk} f_{ijk}(X_i, X_j, \mathbf{E}) \equiv \sum_{(i,j) \in T \setminus (r,s)} \sum_k w_{ijk} f_{ijk}(X_i, X_j, \mathbf{E}),$$

and so *for evidence value* $\mathbf{E}$, the model (1) with edges $T$ is equivalent to (1) with $(r - s)$ removed from $T$. The following notion of *effective CRF structure,* captures the extra sparsity:

**Definition 2** *Given the CRF model* (1) *and evidence value* $\mathcal{E} = \mathbf{E}$, *the effective conditional model structure* $T(\mathcal{E} = \mathbf{E})$ *is the set of edges corresponding to features that are not identically zero:*
$$T(\mathcal{E} = \mathbf{E}) = \{(i, j) \mid (i, j) \in T, \exists k, \mathbf{x_i}, \mathbf{x_j} \ s.t. \ f_{ijk}(\mathbf{x_i}, \mathbf{x_j}, \mathbf{E}) \neq 0\}.$$

If $T(\mathcal{E})$ has low treewidth for all values of $\mathcal{E}$, inference and parameter learning using the effective structure are tractable, even if *a priori* structure $T$ has high treewidth. Unfortunately, in practice the treewidth of $T(\mathcal{E})$ is usually not much smaller than the treewidth of $T$. Low-treewidth effective structures are rarely used, because treewidth is a *global* property of the graph (even *computing* treewidth is NP-complete [13]), while feature design is a *local* process. In fact, it is the ability to learn optimal weights for a set of mutually correlated features without first understanding the inter-feature dependencies that is the key advantage of CRFs over other PGM formulations. Achieving low treewidth for the effective structures requires elaborate feature design, making model construction very difficult. Instead, in this work, we separate construction of low-treewidth effective structures from feature design and weight learning, to combine the advantages of exact inference and discriminative weights learning, high expressive power of high-treewidth models, and local feature design.

Observe that the CRF definition (1) can be written equivalently as
$$P(\mathcal{X} \mid \mathcal{E}, w) = Z^{-1}(\mathcal{E}, w) \exp \left\{ \sum_{ij} \sum_k w_{ijk} \times (\mathbb{I}((i, j) \in T) \cdot f_{ijk}(X_i, X_j, \mathcal{E})) \right\}. \quad (4)$$
Even though (1) and (4) are equivalent, in (4) the structure of the model is explicitly encoded as multiplicative component of the features. In addition to the feature values $f$, the effective structure of the model is now controlled by the indicator functions $\mathbb{I}(\cdot)$. These indicator functions provide us with a way to control the treewidth of the effective structures independently of the features.

Traditionally, it has been assumed that the *a priori* structure $T$ of a CRF model is fixed. However, such an assumption is not necessary. In this work, we assume that the structure is determined by the evidence $\mathcal{E}$ and some parameters $u : T = T(\mathcal{E}, u)$. The resulting model, which we call a CRF with evidence-specific structure (ESS-CRF), defines a conditional distribution $P(\mathcal{X} \mid \mathcal{E}, w, u)$ as follows

$$P(\mathcal{X} \mid \mathcal{E}, w, u) = Z^{-1}(\mathcal{E}, w, u) \exp \left\{ \sum_{ij} \sum_k w_{ijk} (\mathbb{I}((i, j) \in T(\mathcal{E}, u)) \cdot f_{ijk}(X_i, X_j, \mathcal{E})) \right\}. \quad (5)$$
The dependence of the structure $T$ on $\mathcal{E}$ and $u$ can have different forms. We will provide one example of an algorithm for constructing evidence-specific CRF structures shortly.

ESS-CRFs have an important advantage over the traditional parametrization: in (5) the parameters $u$ that determine the model structure are decoupled from the feature weights $w$. As a result, the problem of structure learning (i.e., optimizing $u$) can be decoupled from feature selection (choosing $f$) and feature weights learning (optimizing $w$). Such a decoupling makes it much easier to guarantee that the effective structure of the model has low treewidth by relegating all the necessary global computation to the structure construction algorithm $T = T(\mathcal{E}, u)$. For any fixed choice of a structure construction algorithm $T(\cdot, \cdot)$ and structure parameters $u$, as long as $T(\cdot, \cdot)$ is guaranteed to return low-treewidth structures, learning optimal feature weights $w^*$ and inference at test time can be done exactly, because Fact 1 directly extends to feature weights $w$ in ESS-CRFs:

---
**Algorithm 1**: Standard CRF approach
---
**1** Define features $f_{ijk}(X_i, X_j, \mathcal{E})$, implicitly defining the **high-treewidth** CRF structure $T$.

**2** Optimize weights $w$ to maximize conditional LLH (2) of the training data.
 Use **approximate inference** to compute CLLH objective (2) and gradient (3).

**3 foreach E** *in test data* **do**

**4** | Use conditional model (1) to define the conditional distribution $P(\mathcal{X} \mid \mathbf{E}, w)$.
 | Use **approximate inference** to compute the marginals or the most likely assignment to $\mathcal{X}$.
---

---
**Algorithm 2**: CRF with evidence-specific structures approach
---
**1** Define features $f_{ijk}(X_i, X_j, \mathcal{E})$.
 Choose structure learning alg. $T(\mathcal{E}, u)$ that is guaranteed to return low-treewidth structures.
 Define or learn from data parameters $u$ for the structure construction algorithm $T(\cdot, \cdot)$.

**2** Optimize weights $w$ to maximize conditional LLH $\log P(\mathbf{X} \mid \mathbf{E}, u, w)$ of the training data.
 Use **exact inference** to compute CLLH objective (2) and gradient (3).

**3 foreach E** *in test data* **do**

**4** | Use conditional model (5) to define the conditional distribution $P(\mathcal{X} \mid \mathbf{E}, w, u)$.
 | Use **exact inference** to compute the marginals or the most likely assignment to $\mathcal{X}$.
---

**Observation 3** *Conditional log-likelihood* $\log P(\mathcal{X} \mid \mathcal{E}, w, u)$ *of ESS-CRFs* (5) *is concave in* $w$. *Also,*

$$\frac{\partial \log P(\mathbf{X} \mid \mathbf{E}, w, u)}{\partial w_{ijk}} = \mathbb{I}((i,j) \in T(\mathcal{E}, u))\big(f_{ijk}(\mathbf{X_i}, \mathbf{X_j}, \mathbf{E}) - \mathbb{E}_{P(X_i, X_j \mid \mathbf{E}, w, u)}\left[f_{ijk}(X_i, X_j, \mathbf{E})\right]\big). \quad (6)$$

To summarize, instead of the standard CRF workflow (Alg. 1), we propose ESS-CRFs (Alg. 2). The standard approach has approximations (with little, if any, guarantees on the result quality) at every stage (lines 1,2,4), while in our ESS-CRF approach only structure selection (line 1) involves an approximation. Next, we present a simple but effective algorithm for learning evidence-specific tree structures, based on an existing algorithm for generative models. Many other existing structure learning algorithms can be similarly adapted to learn evidence-specific models of higher treewidth.

## 4  Conditional Chow-Liu algorithm for tractable evidence-specific structures

Learning the most likely PGM structure from data is in most cases intractable. Even for Markov random fields (MRFs), which are a special case of CRFs with no evidence, learning the most likely structure is NP-hard (c.f. [8]). However, for one very simple class of MRFs, namely tree-structured models, an efficient algorithm exists [7] that finds the most likely structure. In this section, we adapt this algorithm (called the Chow-Liu algorithm) to learning evidence-specific structures for CRFs.

Pairwise Markov random fields are graphical models that define a distribution over $\mathcal{X}$ as a normalized product of low-dimensional potentials: $P(\mathcal{X}) \equiv Z^{-1} \prod_{(i,j) \in T} \psi(X_i, X_j)$, Notice that pairwise MRFs are a special case of CRFs with $f_{ij} = \log \psi_{ij}$, $w_{ij} = 1$ and $\mathcal{E} = \emptyset$. Unlike tree CRFs, however, likelihood of tree MRF structures decomposes into contributions of individual edges:

$$LLH(T) = \sum_{(i,j) \in T} I(X_i, X_j) - \sum_{X_i \in \mathcal{X}} H(X_i), \quad (7)$$

where $I(\cdot, \cdot)$ is the mutual information and $H(\cdot)$ is entropy. Therefore, as shown in [7], the most likely structure can be obtained by taking the maximum spanning tree of a fully connected graph, where the weight of an edge $ij$ is $I(X_i, X_j)$. Pairwise marginals have relatively low dimensionality, so the marginals and corresponding mutual informations can be estimated from data accurately, which makes Chow-Liu algorithm a useful one for learning tree-structured models.

Given the concrete value $\mathbf{E}$ of evidence $\mathcal{E}$, one can write down the conditional version of the tree structure likelihood (7) *for that particular value of evidence:*

$$LLH(T \mid \mathbf{E}) = \sum_{(i,j) \in T} I_{P(\cdot \mid \mathbf{E})}(X_i, X_j) - \sum_{X_i \in \mathcal{X}} H_{P(\cdot \mid \mathbf{E})}(X_i). \quad (8)$$

If exact conditional distributions $P(X_i, X_j \mid \mathcal{E})$ were available, then the same Chow-Liu algorithm would find the optimal conditional structure. Unfortunately, estimating conditional distributions $P(X_i, X_j \mid \mathcal{E})$ with fixed accuracy in general requires the amount of data exponential in the dimensionality of $\mathcal{E}$ [14]. However, we can still plug in approximate conditionals $\widehat{P}(\cdot \mid \mathcal{E})$ learned from

---

**Algorithm 3**: Conditional Chow-Liu algorithm for learning evidence-specific tree structures

---

// Parameter learning stage. $u^*$ is found e.g. using L-BFGS with $\widehat{P}(\cdot)$ as in (9)

**1** **foreach** $X_i, X_j \in \mathcal{X}$ **do** $u_{ij}^* \leftarrow \arg\max \sum_{(\mathbf{X},\mathbf{E}) \in \mathcal{D}_{\text{train}}} \log \widehat{P}(\mathbf{X_i}, \mathbf{X_j} \mid \mathbf{E}, u_{ij})$

// Constructing structures at test time

**2** **foreach** $\mathbf{E} \in \mathcal{D}_{test}$ **do**

**3**     **foreach** $X_i, X_j \in \mathcal{X}$ **do** set edge weight $r_{ij}(\mathbf{E}, u_{ij}^*) \leftarrow I_{\widehat{P}(X_i, X_j \mid \mathbf{E}, u_{ij}^*)}(X_i, X_j)$

**4**     $T(\mathbf{E}, u^*) \leftarrow$ maximum spanning tree$(r(\mathbf{E}, u^*))$

---

**Algorithm 4**: Relational ESS-CRF algorithm - parameter learning stage

---

**1** Learn structure parameters $u^*$ using conditional Chow-Liu algorithm (Alg. 3)

**2** Let $P(\mathcal{X} \mid \mathcal{E}, \mathcal{R}, w, u)$ be defined as in (11)

**3** $w^* \leftarrow \arg\max_w \log \widehat{P}(\mathbf{X} \mid \mathbf{E}, \mathcal{R}, w, u^*)$ // Find e.g. with L-BFGS using the gradient (12)

---

data using any standard density estimation technique[3] In particular, with the same features $f_{ijk}$ that are used in the CRF model, one can train a logistic regression model for $\widehat{P}(\cdot \mid \mathcal{E})$ :

$$\widehat{P}(X_i, X_j \mid \mathcal{E}, u_{ij}) = Z_{ij}^{-1}(\mathcal{E}, u_{ij}) \exp\left\{\sum_k u_{ijk} f_{ijk}(X_i, X_j, \mathcal{E})\right\}. \qquad (9)$$

Essentially, a logistic regression model is a small CRF over only two variables. Exact optimal weights $u^*$ can be found efficiently using standard convex optimization techniques.

The resulting evidence-specific structure learning algorithm $T(\mathcal{E}, u)$ is summarized in Alg 3. Alg 3 always returns a tree, and the better the quality of the estimators (9), the better the quality of the resulting structures. Importantly, Alg. 3 is by no means the only choice for the ESS-CRF approach. Other edge scores, e.g. from [4], and edge selection procedures, e.g. [8, 15] for higher treewidth junction trees, can be used as components in the same way as Chow-Liu algorithm is used in Alg. 3.

## 5 Relational CRFs with evidence-specific sructure

Traditional (also called propositional) PGMs are not well suited for dealing with relational data, where every variable is an *entity* of some *type*, and entities are related to each other via different types of links. Usually, there are relatively few entity types and link types. For example, the webpages on the internet are linked via hyperlinks, and social networks link people via friendship relationships. Relational data violates the *i.i.d.* data assumption of traditional PGMs, and huge dimensionalities of relational datasets preclude learning meaningful propositional models. Instead, several formulations of *relational PGMs* have been proposed [16] to work with relational data, including relational CRFs. The key property of all these formulations is that the model is defined using a few *template potentials* defined on the abstract level of *variable types* and replicated as necessary for concrete entities.

More concretely, in relational CRFs every variable $X_i$ is assigned a type $m_i$ out of the set $\mathcal{M}$ of possible types. A binary relation $\mathbf{R} \in \mathcal{R}$, corresponding to a specific type of link between two variables, specifies the types of its input arguments, and a set of features $f_k^{\mathbf{R}}(\cdot, \cdot, \mathcal{E})$ and feature weights $w_k^{\mathbf{R}}$. We will write $X_i, X_j \in \text{inst}(\mathbf{R}, \mathcal{X})$ if the types of $X_i$ and $X_j$ match the input types specified by the relation $\mathbf{R}$ and there is a link of type $\mathbf{R}$ between $X_i$ and $X_j$ in the data (for example, a hyperlink between two webpages). The conditional distribution $P(\mathcal{X} \mid \mathcal{E})$ is then generalized from the propositional CRF (1) by copying the template potentials for every instance of a relation:

$$P(\mathcal{X} \mid \mathcal{E}, \mathcal{R}, w) = Z^{-1}(\mathcal{E}, w) \exp\left\{\sum_{\mathbf{R} \in \mathcal{R}} \sum_{X_i, X_j \in \text{inst}(\mathbf{R}, \mathcal{X})} \sum_k w_k^{\mathbf{R}} f_k^{\mathbf{R}}(X_i, X_j, \mathcal{E})\right\} \quad (10)$$

Observe that the only meaningful difference of the relational CRF (10) from the propositional formulation (1) is that the former shares the same parameters between different edges. By accounting for parameter sharing, it is straightforward to adapt our ESS-CRF formulation to the relational setting. We define the relational ESS-CRF conditional distribution as

$$P(\mathcal{X} \mid \mathcal{E}, \mathcal{R}, w, u) \propto \exp\left\{\sum_{\mathbf{R} \in \mathcal{R}} \sum_{X_i, X_j \in \text{inst}(\mathbf{R}, \mathcal{X})} I((i,j) \in T(\mathcal{E}, u)) \sum_k w_k^{\mathbf{R}} f_k^{\mathbf{R}}(X_i, X_j, \mathcal{E})\right\} \quad (11)$$

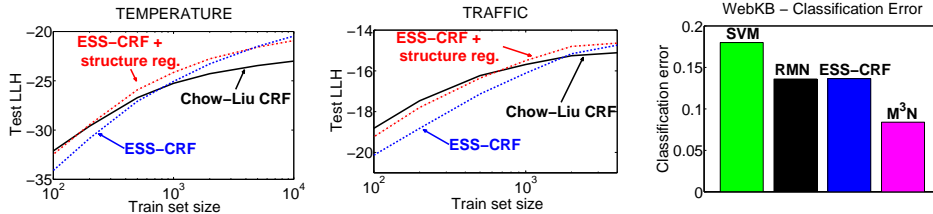

Figure 1: Left: test LLH for TEMPERATURE. Middle: TRAFFIC. Right: classification errors for WebKB.

Given the structure learning algorithm $T(\cdot,\cdot)$ that is guaranteed to return low-treewidth structures, one can learn optimal feature weights $w^*$ and perform inference at test time exactly:

**Observation 4** *Relational ESS-CRF log-likelihood is concave with respect to $w$. Moreover,*

$$\frac{\partial \log P(\mathbf{X} \,|\, \mathbf{E}, \mathcal{R}, w, u)}{\partial w_k^{\mathbf{R}}} = \mathbb{I}(ij \in T(\mathcal{E}, u)) \sum_{X_i, X_j \in inst(\mathbf{R}, \mathcal{X})} \left( f_k^{\mathbf{R}}(\mathbf{X_i}, \mathbf{X_j}, \mathbf{E}) - \mathbb{E}_{P(\cdot \,|\, \mathbf{E}, \mathcal{R}, w, u)} \left[ f_k^{\mathbf{R}}(X_i, X_j, \mathbf{E}) \right] \right). \quad (12)$$

Conditional Chow-Liu algorithm (Alg. 3) can be also extended to the relational setting by using templated logistic regression weights for estimating edge conditionals. The resulting algorithm is shown as Alg. 4. Observe that the test phase of Alg. 4 is exactly the same as for Alg. 3. In the relational setting, one only needs to learn $O(|\mathcal{R}|)$ parameters, *regardless of the dataset size,* for both structure selection and feature weights, as opposed to $O(|\mathcal{X}|^2)$ parameters for the propositional case. Thus, relational ESS-CRFs are typically much less prone to overfitting than propositional ones.

# 6 Experiments

We have tested the ESS-CRF approach on both propositional and relational data. With the large number of parameters needed for the propositional case ($O(|\mathcal{X}|^2)$), our approach is only practical for cases of abundant data. So our experiments with propositional data serve only to prove the concept, verifying that ESS-CRF can successfully learn a model better than a single tree baseline. In contrast to the propositional settings, in the relational cases the relatively low parameter space dimensionality ($O(|\mathcal{R}|^2)$) almost eliminates the overfitting problem. As a result, on relational datasets ESS-CRF is a very attractive approach in practice. Our experiments show ESS-CRFs comfortably outperforming state of the art high-treewidth discriminative models on several real-life relational datasets.

## 6.1 Propositional models

We compare ESS-CRFs with fixed tree CRFs, where the tree structure learned by the Chow-Liu algorithm using $P(\mathcal{X})$. We used TEMPERATURE sensor network data [17] (52 discretized variables) and San Francisco TRAFFIC data [18] (we selected 32 variables). In both cases, 5 variables were used as evidence $\mathcal{E}$ and the rest as unknowns $\mathcal{X}$. The results are in Fig. 1. We have found it useful to regularize the conditional Chow-Liu (Alg. 3) by only choosing at test time from the edges that have been selected often enough during training. In Fig. 1 we plot results for both regularized (red) and unregularized (blue). One can see that in the limit of plentiful data ESS-CRF does indeed outperform the fixed tree baseline. However, because the space of available models is much larger for ESS-CRF, overfitting becomes an important issue and regularization is important.

## 6.2 Relational models

**Face recognition.** We evaluate ESS-CRFs on two relational models. The first model, called FACES, aims to improve face recognition in collections of related images using information about similarity between different faces in addition to the standard single-face features. The key idea is that whenever two people in different images look similar, they are more likely to be the same person. Our model has a variable $X_i$, denoting the label, for every face blob. Pairwise features $f(X_i, X_j, \mathcal{E})$, based on blob color similarity, indicate how close two faces are in appearance. Single-variable features $f(X_i, \mathcal{E})$ encode information such as the output of an off-the-shelf standalone face classifier or face location within the image (see [19] for details). The model is used in a semi-supervised way: at test time, a PGM is instantiated jointly over the train and test entities, values of the train entities are fixed to the ground truth, and inference finds the (approximately) most likely labels for the test entities.

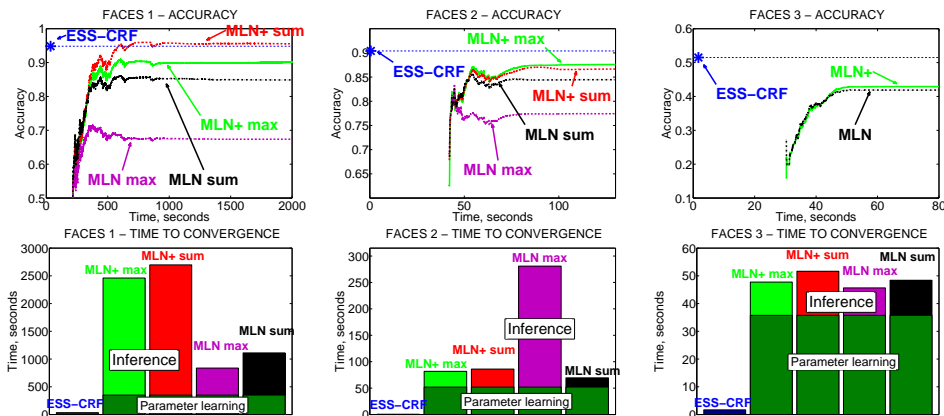

Figure 2: Results for FACES datasets. Top: evolution of classification accuracy as inference progresses over time. Stars show the moment when ESS-CRF finishes running. Horizontal dashed line indicates resulting accuracy. For FACES 3, sum-product and max-product gave the same accuracy. Bottom: time to convergence.

We compare ESS-CRFs with a dense relational PGM encoded by a Markov logic network (MLN, [20]) using the same features. We used a state of the art MLN implementations in the Alchemy package [21] with MC-SAT sampling algorithm for discriminative parameter learning, and belief propagation [22] for inference. For the MLN, we had to threshold the pairwise features indicating the likelihood of label agreement and set those under the threshold to 0 to prevent (a) oversmoothing and (b) very long inference times. Also, to prevent oversmoothing by the MLN, we have found it useful to scale down the pairwise feature weights learned during training, thus weakening the smoothing effect of any single edge in the model[4]. We denote models with so adjusted weights as MLN+. No thresholding or weights adjustment was done for ESS-CRFs.

Figure 2 shows the results on three separate datasets: FACES 1 with 1720 images, 4 unique people and 100 training images in every fold, FACES 2 with 245 images, 9 unique people and 50 training images, and FACES 3 with 352 images, 24 unique people and 70 training images. We tried both sum-product and max-product BP for inference, denoted as sum and max correspondingly in Fig. 2. For ESS-CRF the choice made no difference. One can see that (a) ESS-CRF model provides *superior* (FACES 2 and 3) *or equal* (FACES 1) *accuracy* to the dense MLN model, even with extra heuristic weights tweaking for the MLN, (b) ESS-CRF is *more than an order of magnitude faster*. One can see that for the FACES model, ESS-CRF is clearly superior to the high-treewidth alternative.

**Hypertext data.** For WebKB data (see [23] for details), the task is to label webpages from four computer science departments as `course`, `faculty`, `student`, `project`, or `other`, given their text and link structure. We compare ESS-CRFs to high-treewidth relational Markov networks (RMNs, [23]), max-margin Markov networks (M3Ns, [24]) and a standalone SVM classifier. All the relational PGMs use the same single-variable features encoding the webpage text, and pairwise features encoding the link structure. The baseline SVM classifier only uses single-variable features. RMNs and ESS-CRFs are trained to maximize the conditional likelihood of the labels, while M3Ns maximize the *margin* in likelihood between the correct assignment and all of the incorrect ones, explicitly targeting the classification. The results are in Fig. 1. Observe that ESS-CRF *matches the accuracy of high-treewidth RMNs*, again showing that the smaller expressive power of tree models can be fully compensated by exact parameter learning and inference. ESS-CRF is *much faster* than the RMN, taking only 50 sec. to train and 0.3 sec. to test on a single core of a 2.7GHz Opteron CPU. RMN and M3N models take about 1500 sec. each to train on a 700MHz Pentium III. Even accounting for the CPU speed difference, the speedup is significant. ESS-CRF does not achieve the accuracy of M3Ns, which use a different objective more directly related to the classification problem as opposed to density estimation. Still, the RMN results indicate that it may be possible to match the M3N accuracy with much faster tractable ESS models by replacing the CRF conditional likelihood objective with the max-margin objective, which is an important direction of future work.

# 7 Related work and conclusions

**Related work.** Two cornerstones of our ESS-CRF approach, namely using models that become more sparse when evidence is instantiated, and using *multiple* tractable models to avoid restrictions on the expressive power inherent to low-treewidth models, have been discussed in the existing literature. First, context-specific independence (CSI, [25]) has been long used both for speeding up inference [25] and regularizing the model parameters [26]. However, so far CSI has been treated as a *local* property of the model, which made reasoning about the resulting treewidth of evidence-specific models impossible. Thus, the full potential of *exact* inference for models with CSI remained unused. Our work is a step towards fully exploiting that potential. Multiple tractable models, such as trees, are widely used as components of *mixtures* (e.g. [27]), including mixtures of all possible trees [28], to approximate distributions with rich inherent structure. Unlike the mixture models, our approach of selecting a *single* structure for any given evidence value has the advantage of allowing for efficient exact decoding of the most probable assignment to the unknowns $\mathcal{X}$ using the Viterbi algorithm [29]. Both for the mixture models and our approach, joint optimization of the structure and weights ($u$ and $w$ in our notation) is infeasible due to many local optima of the objective. Our one-shot structure learning algorithm, as we empirically demonstrated, works well in practice. It is also much faster then expectation maximization [30] - the standard way to train mixture models.

Learning the CRF structure in general is NP-hard, which follows from the hardness results for the generative models (c.f. [8]). Moreover, CRF structure learning is further complicated by the fact the CRF structure likelihood does not decompose into scores of local graph components, as do scores for some generative models [3]. Existing work on CRF structure learning thus provides only local guarantees. In practice, the hardness of CRF structure learning leads to high popularity of heuristics: chain and skip-chain [32] structures are often used, as well as grid-like structures. All the approaches that do learn structure from data can be broadly divided into three categories. First, the CRF structure can be defined via the sparsity pattern of the feature weights, so one can use $L1$ regularization penalty to achieve sparsity during weight learning [2]. The second type of approaches greedily adds the features to the CRF model so as to maximize the immediate improvement in the (approximate) model likelihood (e.g. [31]). Finally, one can try to approximate the CRF structure score as a combination of local scores [15, 4] and use an algorithm for learning generative structures (where the score actually decomposes). ESS-CRF also falls in this category of approaches. Although there are some negative theoretical results about learnability of even the simplest CRF structures using local scores [4], such approaches often work well in practice [15].

Learning the weights is straightforward for tractable CRFs, because the log-likelihood is concave [1] and the gradient (3) can be used with mature convex optimization techniques. So far, exact weights learning was mostly used for special hand-crafted structures, such as chains [1, 32], but in this work we use arbitrary trees. For dense structures, computing the gradient (3) exactly is intractable as even approximate inference in general models is NP-hard [5]. As a result, approximate inference techniques, such as belief propagation [10, 11] or Gibbs sampling [12] are employed, without guarantees on the quality of the result. Alternatively, an approximation of the objective (e.g. [6]) is used, also yielding suboptimal weights. Our experiments showed that exact weight learning for tractable models gives an advantage in approximation quality and efficiency over dense structures.

**Conclusions and future work.** To summarize, we have shown that in both propositional and relational settings, tractable CRFs with evidence-specific structures, a class of models with expressive power greater than any single tree-structured model, can be constructed by relying only on the globally optimal results of efficient algorithms (logistic regression, Chow-Liu algorithm, exact inference in tree-structured models, L-BFGS for convex differentiable functions). Whereas traditional CRF workflow (Alg. 1) involves approximation without any quality guaranteed on multiple stages of the process, our approach, ESS-CRF (Alg. 2), has just one source of approximation, namely conditional structure scores. We have demonstrated on real-life relational datasets that our approach matches or exceeds the accuracy of state of the art dense discriminative models, and at the same time provide more than a factor of magnitude speedup. Important future work directions are generalizing ESS-CRF to larger treewidths and max-margin weights learning for better classification.

**Acknowledgements.** This work is supported by NSF Career IIS-0644225 and ARO MURI W911NF0710287 and W911NF0810242. We thank Ben Taskar for sharing the WebKB data. FACES model and data were developed jointly with Denver Dash and Matthai Philipose.

## Footnotes

[1]Here and in the rest of the paper, by "exact parameter learning" we will mean "with arbitrary accuracy in polynomial time" using standard convex optimization techniques. This is in contrast to *closed form* exact parameter learning possible for *generative* low-treewidth models representing the joint distribution $P(\mathcal{X},\mathcal{E})$.

[2]In this paper, we only consider the case of pairwise dependencies, that is, features $f$ that depend on at most two variables from $\mathcal{X}$ (but may depend on arbitrary many variables from $\mathcal{E}$). Our approach can be in principle extended to CRFs with higher order dependencies, but Chow-Liu algorithm for structure learning will have to be replaced with an algorithm that learns low-treewidth junction trees, such as [8].

[3]Notice that the approximation error from $\widehat{P}(\cdot)$ is the *only* source of approximations in all our approach.

[4]Because the number of pairwise relations in the model grows quadratically with the number of variables, the "per-variable force of smoothing" grows with the dataset size, hence the need to adjust.

# References

[1] J. D. Lafferty, A. McCallum, and F. C. N. Pereira. Conditional random fields: Probabilistic models for segmenting and labeling sequence data. In *ICML*, 2001.

[2] M. Schmidt, K. Murphy, G. Fung, and R. Rosales. Structure learning in random fields for heart motion abnormality detection. In *CVPR*, 2008.

[3] D. Koller and N. Friedman. *Probabilistic Graphical Models: Principles and Techniques*. 2009.

[4] J. K. Bradley and C. Guestrin. Learning tree conditional random fields. In *ICML, to appear*, 2010.

[5] D. Roth. On the hardness of approximate reasoning. *Artificial Intelligence*, 82(1-2), 1996.

[6] C. Sutton and A. McCallum. Piecewise pseudolikelihood for efficient CRF training. In *ICML*, 2007.

[7] C. Chow and C. Liu. Approximating discrete probability distributions with dependence trees. *IEEE Trans. on Inf. Theory*, 14(3), 1968.

[8] D. Karger and N. Srebro. Learning Markov networks: Maximum bounded tree-width graphs. In *SODA'01*.

[9] D. C. Liu and J. Nocedal. On the limited memory BFGS method for large scale optimization. *Mathematical Programming*, 45(3), 1989.

[10] J. Pearl. *Probabilistic reasoning in intelligent systems: networks of plausible inference*. 1988.

[11] J. S. Yedidia, W. T. Freeman, and Y. Weiss. Generalized belief propagation. In *NIPS*, 2000.

[12] S. Geman and D. Geman. Stochastic relaxation, Gibbs distributions, and the Bayesian restoration of images. *Pattern Analysis and Machine Intelligence, IEEE Transactions on*, PAMI-6(6), 1984.

[13] S. Arnborg, D. G. Corneil, and A. Proskurowski. Complexity of finding embeddings in a k-tree. *SIAM Journal on Algebraic and Discrete Methods*, 8(2), 1987.

[14] W. Härdle, M. Müller, S. Sperlich, and A. Werwatz. *Nonparametric and Semiparametric Models*. 2004.

[15] D. Shahaf, A. Chechetka, and C. Guestrin. Learning thin junction trees via graph cuts. In *AISTATS*, 2009.

[16] L. Getoor and B. Taskar. *Introduction to Statistical Relational Learning*. The MIT Press, 2007.

[17] A. Deshpande, C. Guestrin, S. Madden, J. Hellerstein, and W. Hong. Model-driven data acquisition in sensor networks. In *VLDB*, 2004.

[18] A. Krause and C. Guestrin. Near-optimal nonmyopic value of information in graphical models. In *UAI'05*.

[19] A. Chechetka, D. Dash, and M. Philipose. Relational learning for collective classification of entities in images. In *AAAI Workshop on Statistical Relational AI*, 2010.

[20] M. Richardson and P. Domingos. Markov logic networks. *Machine Learning*, 62(1-2), 2006.

[21] S. Kok, M. Sumner, M. Richardson, P. Singla, H. Poon, D. Lowd, and P. Domingos. The alchemy system for statistical relational AI. Technical report, University of Washington, Seattle, WA., 2009.

[22] J. Gonzalez, Y. Low, and C. Guestrin. Residual splash for optimally parallelizing belief propagation. In *AISTATS*, 2009.

[23] B. Taskar, P. Abbeel, and D. Koller. Discriminative probabilistic models for relational data. In *UAI*, 2002.

[24] B. Taskar, C. Guestrin, and D. Koller. Max-margin markov networks. In *NIPS*, 2003.

[25] C. Boutilier, N. Friedman, M. Goldszmidt, and D. Koller. Context-specific independence in Bayesian networks. In *UAI*, 1996.

[26] M. desJardins, P. Rathod, and L. Getoor. Bayesian network learning with abstraction hierarchies and context-specific independence. In *ECML*, 2005.

[27] B. Thiesson, C. Meek, D. Chickering, and D. Heckerman. Learning mixtures of DAG models. In *UAI'97*.

[28] M. Meilă and M. I. Jordan. Learning with mixtures of trees. *JMLR*, 1, 2001.

[29] A. J. Viterbi. Error bounds for convolutional codes and an asymptotically optimum decoding algorithm. *IEEE Transactions on Information Theory*, IT-13, 1967.

[30] S. L. Lauritzen. The EM algorithm for graphical association models with missing data. *Computational Statistics & Data Analysis*, 19(2), 1995.

[31] A. Torralba, K. P. Murphy, and W. T. Freeman. Contextual models for object detection using boosted random fields. In *NIPS*, 2004.

[32] C. Sutton and A. McCallum. Collective segmentation and labeling of distant entities in information extraction. In *ICML Workshop on Statistical Relational Learning and Its Connections*, 2004.

